# Submodularity Cuts and Applications

**Yoshinobu Kawahara**[*]
The Inst. of Scientific and Industrial Res. (ISIR),
Osaka Univ., Japan
kawahara@ar.sanken.osaka-u.ac.jp

**Kiyohito Nagano**
Dept. of Math. and Comp. Sci.,
Tokyo Inst. of Technology, Japan
nagano@is.titech.ac.jp

**Koji Tsuda**
Comp. Bio. Research Center,
AIST, Japan
koji.tsuda@aist.go.jp

**Jeff A. Bilmes**
Dept. of Electrical Engineering,
Univ. of Washington, USA
bilmes@u.washington.edu

## Abstract

Several key problems in machine learning, such as feature selection and active learning, can be formulated as submodular set function maximization. We present herein a novel algorithm for maximizing a submodular set function under a cardinality constraint — the algorithm is based on a cutting-plane method and is implemented as an iterative small-scale binary-integer linear programming procedure. It is well known that this problem is **NP**-hard, and the approximation factor achieved by the greedy algorithm is the theoretical limit for polynomial time. As for (non-polynomial time) exact algorithms that perform reasonably in practice, there has been very little in the literature although the problem is quite important for many applications. Our algorithm is guaranteed to find the exact solution finitely many iterations, and it converges fast in practice due to the efficiency of the cutting-plane mechanism. Moreover, we also provide a method that produces successively decreasing upper-bounds of the optimal solution, while our algorithm provides successively increasing lower-bounds. Thus, the accuracy of the current solution can be estimated at any point, and the algorithm can be stopped early once a desired degree of tolerance is met. We evaluate our algorithm on sensor placement and feature selection applications showing good performance.

## 1 Introduction

In many fundamental problems in machine learning, such as feature selection and active learning, we try to select a subset of a finite set so that some utility of the subset is maximized. A number of such utility functions are known to be submodular, i.e., the set function $f$ satisfies $f(S) + f(T) \geq f(S \cap T) + f(S \cup T)$ for all $S, T \subseteq V$, where $V$ is a finite set [2, 5]. This type of function can be regarded as a discrete counterpart of convex functions, and includes entropy, symmetric mutual information, information gain, graph cut functions, and so on. In recent years, treating machine learning problems as submodular set function maximization (usually under some constraint, such as limited cardinality) has been addressed in the community [10, 13, 22].

In this paper, we address submodular function maximization under a cardinality constraint:

$$\max_{S \subseteq V} f(S) \quad \text{s.t.} \ |S| \leq k, \tag{1}$$

where $V = \{1, 2, \ldots, n\}$ and $k$ is a positive integer with $k \leq n$. Note that this formulation is considerably general and covers a broad range of problems. The main difficulty of this problem comes from a potentially exponentially large number of locally optimal solutions. In the field of

---

[*]URL: http://www.ar.sanken.osaka-u.ac.jp/ kawahara/

combinatorial optimization, it is well-known that submodular maximization is **NP**-hard and the approximation factor of $(1 - 1/e)$ ($\approx 0.63$) achieved by the greedy algorithm [19] is the theoretical limit of a polynomial-time algorithm for positive and nondecreasing submodular functions [3]. That is, in the worst case, any polynomial-time algorithm cannot give a solution whose function value is more than $(1 - 1/e)$ times larger than the optimal value unless P=NP. In recent years, it has been reported that greedy-based algorithms work well in several machine-learning problems [10, 1, 13, 22]. However, in some applications of machine learning, one seeks a solution closer to the optimum than what is guaranteed by this bound. In feature selection or sensor placement, for example, one may be willing to spend much more time in the selecting phase, since once selected, items are used many times or for a long duration. Unfortunately, there has been very little in the literature on finding exact but still practical solutions to submodular maximization [17, 14, 8]. To the best of our knowledge, the algorithm by Nemhauser and Wolsey [17] is the only way for exactly maximizing a general form of nondecreasing submodular functions (other than naive brute force). However, as stated below, this approach is inefficient even for moderate problem sizes.

In this paper, we present a novel algorithm for maximizing a submodular set function under a cardinality constraint based on a cutting-plane method, which is implemented as an iterative small-scale binary-integer linear programming (BILP) procedure. To this end, we derive *the submodularity cut*, a cutting plane that cuts off the feasible sets on which the objective function values are guaranteed to be not better than current best one, and this is based on the submodularity of a function and its Lovász extension [15, 16]. This cut assures convergence to the optimum in finite iterations and allows the searching for better subsets in an efficient manner so that the algorithm can be applied to suitably-sized problems. The existing algorithm [17] is infeasible for such problems since, as originally presented, it has no criterion for improving the solution efficiently at each iteration (we compare these algorithms empirically in Sect. 5.1). Moreover, we present a new way to evaluate an upper bound of the optimal value with the help of the idea of Nemhauser and Wolsey [17]. This enables us to judge the accuracy of the current best solution and to calculate an $\epsilon$-optimal solution for a predetermined $\epsilon > 0$ (cf. Sect. 4). In our algorithm, one needs to iteratively solve small-scale BILP (and mixed integer programming (MIP) for the upper-bound) problems, which are also **NP**-hard. However, due to their small size, these can be solved using efficient modern software packages such as CPLEX. Note that BILP is a special case of MIP and more efficient to solve in general, and the presented algorithm can be applied to any submodular functions while the existing one needs the nondecreasing property.[1] We evaluate the proposed algorithm on the applications of sensor placement and feature selection in text classification.

The remainder of the paper is organized as follows: In Sect. 2, we present submodularity cuts and give a general description of the algorithm using this cutting plane. Then, we describe a specific procedure for performing the submodularity cut algorithm in Sect. 3 and the way of updating an upper bound for calculating an $\epsilon$-optimal solution in Sect. 4. And finally, we give several empirical examples in Sect. 5, and conclude the paper in Sect. 6.

## 2  Submodularity Cuts and Cutting-Plane Algorithm

We start with a subset $S_0 \subseteq V$ of some ground set $V$ with a reasonably good lower bound $\gamma = f(S_0) \leq \max\{f(S) : S \subseteq V\}$. Using this information, we cut off the feasible sets on which the objective function values are guaranteed to be not better than $f(S_0)$. In this section, we address a method for solving the submodular maximization problem (1) based on this idea along the line of cutting-plane methods, as described by Tuy [23] (see also [6, 7]) and often successfully used in algorithms for solving mathematical programming problems [18, 11, 20].

### 2.1  Lovász extension

For dealing with the submodular maximization problem (1) in a way analogous to the continuous counterpart, i.e., convex maximization, we briefly describe an useful extension to submodular functions, called the Lovász extension [15, 16]. The relationship between the discrete and the continuous, described in this subsection, is summarized in Table 1.

Table 1: Correspondence between continuous and discrete.

| (discrete) | | (continuous) |
|---|---|---|
| $f : 2^V \to \mathbb{R}$ | $\overset{\text{Eq. (2)}}{\Longrightarrow}$ | $\hat{f} : \mathbb{R}^n \to \mathbb{R}$ |
| $S \subseteq V$ | $\overset{\text{Eq. (3)}}{\Longleftrightarrow}$ | $\boldsymbol{I}_S \in \mathbb{R}^n$ |
| $f$ is submodular | $\overset{\text{Thm. 1}}{\Longleftrightarrow}$ | $\hat{f}$ is convex |

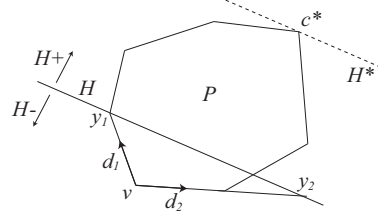

Figure 1: Illustration of cutting plane $H$. For $H^*$ and $c^*$, see Section 3.2.

Given any real vector $\boldsymbol{p} \in \mathbb{R}^n$, we denote the $m$ distinct elements of $\boldsymbol{p}$ by $\hat{p}_1 > \hat{p}_2 > \cdots > \hat{p}_m$. Then, the Lovász extension $\hat{f} : \mathbb{R}^n \to \mathbb{R}$ corresponding to a general set function $f : 2^V \to \mathbb{R}$, which is not necessarily submodular, is defined as

$$\hat{f}(\boldsymbol{p}) = \sum_{k=1}^{m-1}(\hat{p}_k - \hat{p}_{k+1})f(U_k) + \hat{p}_m f(U_m), \tag{2}$$

where $U_k = \{i \in V : p_i \geq \hat{p}_k\}$. From the definition, $\hat{f}$ is a piecewise linear (i.e., polyhedral) function.[2] In general, $\hat{f}$ is not convex. However, the following relationship between the submodularity of $f$ and the convexity of $\hat{f}$ is given [15, 16]:

**Theorem 1** *For a set function $f : 2^V \to \mathbb{R}$ and its Lovász extension $\hat{f} : \mathbb{R}^n \to \mathbb{R}$, $f$ is submodular if and only if $\hat{f}$ is convex.*

Now, we define $\boldsymbol{I}_S \in \{0, 1\}^n$ as $\boldsymbol{I}_S = \sum_{i \in S} \boldsymbol{e}_i$, where $\boldsymbol{e}_i$ is the $i$-th unit vector. Obviously, there is a one-to-one correspondence between $\boldsymbol{I}_S$ and $S$. $\boldsymbol{I}_S$ is called the *characteristic vector* of $S$.[3] Then, the Lovász extension $\hat{f}$ is a natural extension of $f$ in the sense that it satisfies the following [15, 16]:

$$\hat{f}(\boldsymbol{I}_S) = f(S) \quad (S \subseteq V). \tag{3}$$

In what follows, we assume that $f$ is submodular. Now we introduce a continuous relaxation of the problem (1) using the Lovász extension $\hat{f}$. A polytope $P \subseteq \mathbb{R}^n$ is a bounded intersection of a finite set of half-spaces — that is, $P$ is of the form $P = \{\boldsymbol{x} \in \mathbb{R}^n : A_j^\top \boldsymbol{x} \leq b_j, j = 1, \cdots, m\}$, where $A_j$ is a real vector and $b_j$ is a real scalar. According to the correspondence between discrete and continuous functions described above, it is natural to replace the objective function $f : 2^V \to \mathbb{R}$ and the feasible region $\{S \subseteq V : |S| \leq k\}$ of the problem (1) by the Lovász extension $\hat{f} : \mathbb{R}^n \to \mathbb{R}$ and a polytope $D_0 \subseteq \mathbb{R}^n$ defined by

$$D_0 = \{\boldsymbol{x} \in \mathbb{R}^n : 0 \leq x_i \leq 1 \ (i = 1, \cdots, n), \ \textstyle\sum_{i=1}^n x_i \leq k\},$$

respectively. The resulting problem is a convex maximization problem. For problem (1), we will use the analogy with the way of solving the continuous problem: $\max\{\hat{f}(\boldsymbol{x}) : \boldsymbol{x} \in D_0\}$. The question is, can we solve it and how good is the solution?

## 2.2 Submodularity cuts

Here, we derive what we call the *submodularity cut*, a cutting plane that cuts off the feasible sets with optimality guarantees using the submodularity of $f$, and with the help of the relationship between submodularity and convexity described in Thm. 1. Note that the algorithm using this cutting plane, described later, converges to an optimal solution in a finite number of iterations (cf. Thm. 5). The presented technique is essentially a discrete analog of concavity cut techniques for continuous concave minimization, which rests on the following property (see, e.g., [11]).

**Theorem 2** *A convex function $g : \mathbb{R}^n \to \mathbb{R}$ attains its global maximum over a polytope $P \subset \mathbb{R}^n$ at a vertex of $P$.*

$$\hat{f}(\boldsymbol{p}) = \sup\{\boldsymbol{p}^T \boldsymbol{x} : \boldsymbol{x} \in \boldsymbol{B}(f)\} \quad (\boldsymbol{p} \in \mathbb{R}^n),$$

where $\boldsymbol{B}(f) = \{\boldsymbol{x} \in \mathbb{R}^n : \boldsymbol{x}(S) \leq f(S) \ (\forall S \subset V), \boldsymbol{x}(V) = f(V)\}$ is the base polyhedron associated with $f$ [15] and $\boldsymbol{x}(S) = \sum_{i \in S} x_i$.

[3]For example in case of $|V| = 6$, the characteristic vector of $S = \{1, 3, 4\}$ becomes $\boldsymbol{I}_S = (1, 0, 1, 1, 0, 0)$.

First, we clarify the relation between discrete and continuous problems. Let $P$ be a polytope with $P \subseteq D_0$. Denote by $S(P)$ the subsets of $V$ whose characteristic vectors are inside of $P$, i.e., $\boldsymbol{I}_{S'} \in P$ for any $S' \in S(P)$, and denote by $V(P)$ the set consisting of all vertices of $P$. Note that any characteristic vector $\boldsymbol{I}_S \in P$ is a vertex of $P$. Also, there is a one-to-one correspondence between $S(D_0)$ and $V(D_0)$. Now clearly, we have

$$\max\{f(S') : S' \in S(P)\} \leq \max\{\hat{f}(\boldsymbol{x}) : \boldsymbol{x} \in P\}. \tag{4}$$

If we can find a subset $\bar{P}$ where the function value of $\hat{f}$ is always smaller than the currently-known largest value, any $f(\bar{S})$ for $\bar{S} \in S(\bar{P})$ is also smaller than the value. Thus, the cutting plane for the problem $\max\{\hat{f}(\boldsymbol{x}) : \boldsymbol{x} \in D_0\}$ can be applied to our problem (1) through the relationship (4).

To derive the submodularity cut, we use the following definition:

**Definition 3** ($\gamma$-extension) *Let $g : \mathbb{R}^n \to \mathbb{R}$ be a convex function, $\boldsymbol{x} \in \mathbb{R}^n$, $\gamma$ be a real number satisfying $\gamma \geq g(\boldsymbol{x})$ and $t > 0$. Then, a point $\boldsymbol{y} \in \mathbb{R}^n$ defined by the following formula is called $\gamma$-extension of $\boldsymbol{x}$ in direction $\boldsymbol{d} \in \mathbb{R}^n \setminus \{0\}$ (with respect to g) where $\theta \in \mathbb{R} \cup \{\infty\}$:*

$$\boldsymbol{y} = \boldsymbol{x} + \theta\boldsymbol{d} \quad with \quad \theta = \sup\{t : g(\boldsymbol{x} + t\boldsymbol{d}) \leq \gamma\}. \tag{5}$$

We may have $\theta = \infty$ depending on $g$ and $\boldsymbol{d}$, but this is unproblematic in practice. The $\gamma$-extension of $x \in \mathbb{R}^n$ can be defined with respect to the Lovász extension because it is a convex function.

The submodular cut algorithm is an iterative procedure. At each iteration, the algorithm keeps a polytope $P \subseteq D_0$, the current best function value $\gamma$, and a set $S^* \subseteq V$ satisfying $f(S^*) = \gamma$. We construct a submodular cut as follows. Let $\boldsymbol{v} \in V(P)$ be a vertex of $P$ such that $v = \boldsymbol{I}_S$ for some $S \in S(P)$, and let $K = K(\boldsymbol{v}; \boldsymbol{d}_1, \ldots, \boldsymbol{d}_n)$ be a convex polyhedral cone with vertex $\boldsymbol{v}$ generated by linearly independent vectors $\boldsymbol{d}_1, \ldots, \boldsymbol{d}_n$, i.e., $K = \{\boldsymbol{v} + t_1\boldsymbol{d}_1 + \cdots + t_n\boldsymbol{d}_n : t_l \geq 0\}$. For each $i = 1, \cdots, n$, let $\boldsymbol{y}_l = \boldsymbol{v} + \theta_l\boldsymbol{d}_l$ be the $\gamma$-extension of $\boldsymbol{v}$ in direction $\boldsymbol{d}_l$ with respect to $\hat{f}$. We choose the vectors $\boldsymbol{d}_1, \ldots, \boldsymbol{d}_n$ so that $P \subset K$ and $\theta_l > 0$ (cf. Sect. 3.1). These directions are not necessarily chosen tightly on $P$ (in fact, the directions described in Sect. 3.1 enclose $P$ but also a set larger). Since the vectors $\boldsymbol{d}_l$ are linearly independent, there exists a unique hyperplane $H = H(\boldsymbol{y}_1, \cdots, \boldsymbol{y}_n)$ that contains $\boldsymbol{y}_l$ ($l = 1, \cdots, n$), which we call a *submodular cut*. It is defined by (cf. Fig. 1)

$$H = \{\boldsymbol{x} : \boldsymbol{e}^T Y^{-1}\boldsymbol{x} = 1 + \boldsymbol{e}^T Y^{-1}\boldsymbol{v}\}. \tag{6}$$

where $\boldsymbol{e} = (1, \cdots, 1)^T \in \mathbb{R}^n$ and $Y = ((\boldsymbol{y}_1 - \boldsymbol{v}), \cdots, (\boldsymbol{y}_n - \boldsymbol{v}))$. The hyperplane $H$ generates two halfspaces $H_- = \{\boldsymbol{x} : \boldsymbol{e}^T Y^{-1}\boldsymbol{x} \leq 1 + \boldsymbol{e}^T Y\boldsymbol{v}\}$ and $H_+ = \{\boldsymbol{x} : \boldsymbol{e}^T Y^{-1}\boldsymbol{x} \geq 1 + \boldsymbol{e}^T Y\boldsymbol{v}\}$. Obviously the point $\boldsymbol{v}$ is in the halfspace $H_-$, and moreover, we have:

**Lemma 4** *Let $P \subseteq D_0$ be a polytope, $\gamma$ be the current best function value, $\boldsymbol{v}$ be a vertex of $P$ such that $\boldsymbol{v} = \boldsymbol{I}_S$ for some $S \in S(P)$ and $H_-$ be the halfspace determined by the cutting plane, i.e., $H_- = \{\boldsymbol{x} : \boldsymbol{e}^T Y^{-1}\boldsymbol{x} \leq 1 + \boldsymbol{e}^T Y\boldsymbol{v}\}$, where $Y = ((\boldsymbol{y}_1 - \boldsymbol{v}), \cdots, (\boldsymbol{y}_n - \boldsymbol{v}))$ and $\boldsymbol{y}_1, \ldots, \boldsymbol{y}_n$ are the $\gamma$-extensions of $\boldsymbol{v}$ in linearly independent directions $\boldsymbol{d}_1, \ldots, \boldsymbol{d}_n$. Then, it holds that*

$$f(S') \leq \gamma \quad for \ all \ S' \in S(P \cap H_-).$$

**Proof** Since $P \subset K = K(\boldsymbol{I}_S; \boldsymbol{d}_1, \cdots, \boldsymbol{d}_n)$, it follows that $P \cap H_-$ is contained in the simplex $R = [\boldsymbol{I}_S, \boldsymbol{y}_1, \cdots, \boldsymbol{y}_n]$. Since the Lovász extension $\hat{f}$ is convex and the maximum of a convex function over a compact convex set is attained at a vertex of the convex set (Thm. 2), the maximum of $\hat{f}$ over $R$ is attained at a vertex of $R$. Therefore, we have

$$\max\{\hat{f}(\boldsymbol{x}) : \boldsymbol{x} \in P \cap H_-\} \leq \max\{f(\boldsymbol{x}) : \boldsymbol{x} \in R\} = \max\{\hat{f}(\boldsymbol{v}); \hat{f}(\boldsymbol{y}_1), \cdots, \hat{f}(\boldsymbol{y}_n)\} \leq \gamma.$$

From Eq. (4), $\max\{f(S') : S' \in S(P \cap H_-)\} \leq \max\{\hat{f}(\boldsymbol{x}) : \boldsymbol{x} \in P \cap H_-\} \leq \gamma$. ∎

The above lemma shows that we can cut off the feasible subsets $S(P \cap H_-)$ from $S(P)$ without loss of any feasible set whose objective function value is better than $\gamma$. If $S(P) = S(P \cap H_-)$, then $\gamma = \max\{f(S) : |S| \leq k\}$ is achieved. A specific way to check whether $S(P) = S(P \cap H_-)$ will be given in Sect. 3.2. As $\boldsymbol{v} \in S(P \cap H_-)$ and $\boldsymbol{v} \notin S(P \cap H_+)$, we have

$$|S(P)| > |S(P \cap H_+)|. \tag{7}$$

The submodular cut algorithm updates $P \leftarrow P \cap H_+$ until the global optimality of $\gamma$ is guaranteed. The general description is shown in Alg. 1 (also see Fig. 2). Furthermore, the finiteness of the algorithm is assured by the following theorem.

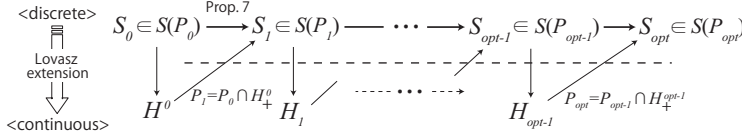

Figure 2: Outline of the submodularity cuts algorithm.

---

**Algorithm 1** General description of the submodularity cuts algorithm.

---

1. Compute a subset $S_0$ s.t. $|S_0| \leq k$, and set a lower bound $\gamma_0 = f(S_0)$.
2. Set $P_0 \leftarrow D_0$, $stop \leftarrow false$, $i \leftarrow 1$ and $S^* = S_0$.
3. **while** stop=false **do**
4.     Construct with respect to $S_{i-1}$, $P_{i-1}$ and $\gamma_{i-1}$ a submodularity cut $H^i$.
5.     **if** $S(P_{i-1}) = S(P_{i-1} \cap H_-^i)$ **then**
6.         $stop \leftarrow true$ ($S^*$ is an optimal solution and $\gamma_{i-1}$ the optimal value).
7.     **else**
8.         Update $\gamma_i$ (using $S_i$ and other available information) and set $S^*$ s.t. $f(S^*) = \gamma_i$.
9.         Compute $S_i \in S(P_i)$, and set $P_i \leftarrow P_{i-1} \cap H_+^i$ and $i \leftarrow i+1$.
10.    **end if**
11. **end while**

---

**Theorem 5** *Alg. 1 gives an optimal solution to the problem* (1) *in a finite number of iterations.*

**Proof** In the beginning, $|S(D_0)|$ is finite. In view of (7), each iteration decreases $|S(P)|$ by at least 1. So, the number of iterations is finite. ∎

# 3 Implementation

In this section, we describe a specific way to perform Alg. 1 using a binary-integer linear programming (BILP) solver. The pseudo-code of the resulting algorithm is shown in Alg. 2.

## 3.1 Construction of submodularity cuts

Given a vertex of a polytope $P \subseteq D_0$, which is of the form $\boldsymbol{I}_S$, we describe how to compute linearly independent directions $\boldsymbol{d}_1, \cdots, \boldsymbol{d}_n$ for the construction of the submodularity cut at each iteration of the algorithm (Line 4 in Alg. 1). Note that the way described here is just one option and any other choice satisfying $P \subset K$ can be substituted.

If $|S| < k$, then directions $\boldsymbol{d}_1, \ldots, \boldsymbol{d}_n$ can be chosen as $-\boldsymbol{e}_l$ ($l \in S$) and $\boldsymbol{e}_l$ ($l \in V \setminus S$). Now we focus on the case where $|S| = k$. Define a neighbor $S_{(i,j)}$ of $S$ as

$$S_{(i,j)} := (S \setminus \{i\}) \cup \{j\} \quad (i \in S, \ j \in V \setminus S).$$

That is, the neighbor $S_{(i,j)}$ is given by replacing one of the elements of $S$ with that of $V \setminus S$. Note that $\boldsymbol{I}_{S_{(i,j)}} - \boldsymbol{I}_S = \boldsymbol{e}_j - \boldsymbol{e}_i$ for any neighbor $S_{(i,j)}$ of $S$. Let $S_{(i^*,j^*)}$ be a neighbor that maximizes $f(S_{(i,j)})$ among all neighbors of $S$. Since a subset $S$ of size $k$ has $k \times (n-k)$ neighbors $S_{(i,j)}$ ($i \in S$, $j \in V \setminus S$), this computation is $O(nk)$. Suppose that $S = \{i_1, \ldots, i_k\}$ with $i_1 = i^*$ and $V \setminus S = \{j_{k+1}, \ldots, j_n\}$ with $j_n = j^*$. If $f(S_{(i^*,j^*)}) > \gamma$, we update $\gamma \leftarrow f(S_{(i^*,j^*)})$ and $S^* \leftarrow S_{(i^*,j^*)}$. Thus, in either case it holds that $\gamma \geq f(S_{(i^*,j^*)})$. As an example of the set of directions $\{\boldsymbol{d}_1, \ldots, \boldsymbol{d}_n\}$, we choose

$$\boldsymbol{d}_l = \begin{cases} \boldsymbol{e}_{j^*} - \boldsymbol{e}_{i_l} & \text{if } l \in \{1, \ldots, k\} \\ \boldsymbol{e}_{j_l} - \boldsymbol{e}_{j^*} & \text{if } l \in \{k+1, \ldots, n-1\} \\ -\boldsymbol{e}_{j^*} & \text{if } l = n. \end{cases} \quad (8)$$

It is easy to see that $\boldsymbol{d}_1, \ldots, \boldsymbol{d}_n$ are linearly independent. Moreover, we obtain the following lemma:

**Lemma 6** *For the directions $\boldsymbol{d}_1, \ldots, \boldsymbol{d}_n$ defined in* (8)*, a cone*
$$K(\boldsymbol{I}_S; \boldsymbol{d}_1, \ldots, \boldsymbol{d}_n) = \{\boldsymbol{I}_S + t_1 \boldsymbol{d}_1 + \cdots + t_n \boldsymbol{d}_n : t_l \geq 0\}$$
*contains the polytope $D_0 = \{\boldsymbol{x} \in \mathbb{R}^n : 0 \leq x_l \leq 1 \ (l = 1, \cdots, n), \ \sum_{l=1}^n x_l \leq k\}$.*

The proof of this lemma is included in the supplementary material (Sect. A). The $\gamma$-extensions, i.e., $\theta$'s, in these directions can be obtained in closed forms. The details of this are also included in the supplementary material (Sect. A).

**Algorithm 2** Pseudo-code of the submodularity cuts algorithm using BILP.

1. Compute a subset $S_0$ s.t. $|S_0| \leq k$, and set a lower bound $\gamma_0 = f(S_0)$.
2. Set $P_0 \leftarrow D_0$, $stop \leftarrow false$, $i \leftarrow 1$ and $S^* = S_0$.
3. **while** stop=false **do**
4.     Construct with respect to $S_{i-1}$, $P_{i-1}$ and $\gamma_{i-1}$ a submodularity cut $H$.
5.     Solve the BILP problem (9) with respect to $A_j$ and $b_j$ ($j = 1, \cdots, n_k$), and let the optimal solution and value $S_i$ and $c^*$, respectively.
6.     **if** $c^* \leq 1 + e^T Y^{-1} v_{i-1}$ **then**
7.         $stop \leftarrow true$ ($S^*$ is an optimal solution and $\gamma_{i-1}$ the optimal value).
8.     **else**
9.         Update $\gamma_i$ (using $S_i$ and other available information) and set $S^*$ s.t. $f(S^*) = \gamma_i$.
10.         Set $P_i \leftarrow P_{i-1} \cap H_+$ and $i \leftarrow i + 1$.
11.     **end if**
12. **end while**

## 3.2 Stopping criterion and next starting point

Next, we address the checking of optimality, i.e., whether $S(P) = S(P \cap H_-)$, and also finding the next starting subset $S_i$ (respectively, in Lines 5 and 9 in Alg. 1). Let $\widetilde{P} \subseteq \mathbb{R}^n$ be the minimum polytope containing $S(P)$. Geometrically, checking $S(P) = S(P \cap H_-)$ can be done by considering a parallel hyperplane $H^*$ of $H$ which is tangent to $\widetilde{P}$. If $H = H^*$ or $H^*$ is given by translating $H$ towards $v$, then $S(P) = S(P \cap H_-)$. Numerically, such a translation corresponds to linear programming. Using Eq. (6), we obtain:

**Proposition 7** *Let $c^*$ be the optimal value of the binary integer program*

$$\max_{x \in \{0,1\}^n} \{e^T Y^{-1} x : A_j x \geq b_j, j = 1, \cdots, m_k\}. \tag{9}$$

*Then $S(P) \subset H_-$ if $c^* \leq 1 + e^T Y^{-1} v$.*

Note that, if $c^* > 1 + e^T Y^{-1} v$, then the optimal solution $x^*$ of Eq. (9) yields a subset of $S(P \setminus H_-)$ which can be used as a starting subset of the next iteration (see Fig. 1).

## 4 Upper bound and $\epsilon$-optimal solution

Although our algorithm can find an exact solution in a finite number of iterations, the computational cost could be expensive for a high-dimensional case. Therefore, we present here an iterative update of an upper bound of the current solution, and thus a way to allow us to obtain an $\epsilon$-optimal solution. To this end, we combine the idea of the algorithm by Nemhauser and Wolsey [17] with our cutting plane algorithm. Note that this hybrid approach is effective only when $f$ is nondecreasing.

If the submodular function $f : 2^V \rightarrow \mathbb{R}$ is nondecreasing, the submodular maximization problem (1) can be reformulated [17] as

$$\max \eta \quad \text{s.t.} \quad \begin{aligned} &\eta \leq f(S) + \sum_{j \in V \setminus S} \rho_j(S) y_j \quad (S \subseteq V), \\ &\sum_{j \in V} y_j = k, \ y_j \in \{0,1\} \ (j \in V) \end{aligned} \tag{10}$$

where $\rho_j(S) := f(S \cup \{j\}) - f(S)$. This formulation is a MIP with regard to one continuous and $n$ binary variables, and has approximately $2^n$ constraints. The first type of constraint corresponds to all feasible subsets $S$, and the number of inequalities is as large as $2^n$. This approach is therefore infeasible for certain problem sizes. Nemhauser and Wolsey [17] address this problem by adding the constraints one by on and calculating a reduced MIP problem iteratively. In the worse case, however, the number of iterations becomes equal to the case of when all constraints are added. The solution of a maximization problem with a subset of constraints is larger than the one with all constraints, so the good news is that this solution is guaranteed to improve (by monotonically decreasing down to the true solution) at each iteration. In our algorithm, by contrast, the best current solution increases monotonically to the true solution. Therefore, by adding the constraint corresponding to $S_i$ at each iteration of our algorithm and solving the reduced MIP above, we can evaluate an upper bound of the current solution. Thus, we can assure the optimality of a current solution, or obtain a desired $\epsilon$-optimal solution using both the lower and upper bound.

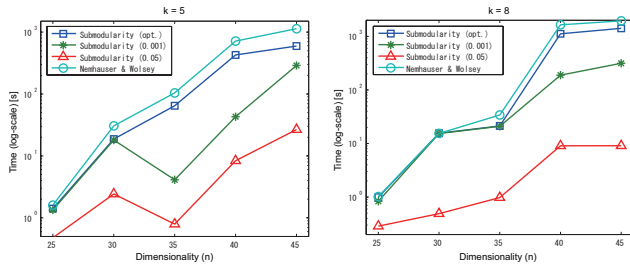
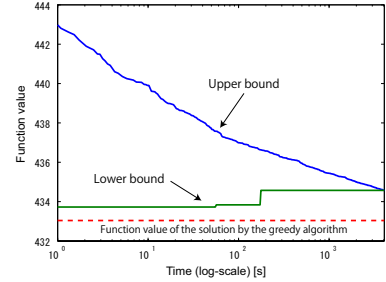

Figure 3: Averaged computational time (log-scale) for computing exact and $\epsilon$-optimal solutions by the submodularity cut algorithm and existing algorithm by Nemhauser and Wolsey.

Figure 4: An example of computational time (log-scale) versus the calculated upper and lower bounds.

## 5 Experimental Evaluation

We first empirically compare the proposed algorithm with the existing algorithm by Nemhauser and Wolsey [17] in Sect. 5.1, and then apply the algorithm to the real-world applications of sensor placement, and feature selection in text classification (Sect. 5.2 and 5.3, respectively). In the experiments, we used the solution by a greedy algorithm as initial subset $S_0$. The experiments below were run on a 2.5GHz 64-bit workstation using Matlab and a Parallel CPLEX ver. 11.2 (8 threads) through a mex function. If $\theta = \infty$ in Eq. (5), we set $\theta = \theta_1$, where $\theta_1$ is large (i.e. $\theta_1 = 10^6$).

### 5.1 Artificial example

Here, we evaluate empirically and illustrate the submodularity cut algorithm (Alg. 2) with respect to (1) computational time for exact solutions compared with the existing algorithm and (2) how fast the algorithm can sandwich the true solution between the upper and lower bounds, using artificial datasets. The considered problem here is the $K$-location problem [17], i.e., the submodular maximization problem (1) with respect to the nondecreasing submodular function:

$$f(S) = \sum_{i=1}^{m} \max_{j \in S} c_{ij},$$

where $C = c_{ij}$ is an $m \times n$ nonnegative matrix and $V = \{1, \cdots, n\}$. We generated several matrices $C$ of different size $n$ (we fixed $m = n+1$), and solved the above problem with respect to $k = 5, 8$ for exact and $\epsilon$ optimal solutions, using the two algorithms. The graphs in Fig. 3 show the computational time (log-scale) for several $n$ and $k = 5, 8$, where the results were averaged over randomly generated 3 matrices $C$. Note that, for example, the number of combination becomes more than two hundred millions for $n = 45$ and $k = 8$. As the figure shows, the required costs for Alg. 2 were less than the existing algorithm, especially in the case of high search spaces. This could be because the cutting-plane algorithm searches feasible subsets in an efficient manner by eliminating worse ones with the submodularity cuts. And Fig. 4 shows an example of the calculated upper and lower bounds vs. time ($k = 5$ and $n = 45$). The lower bound is updated rarely and converges to the optimal solution quickly while the upper bound decreases gradually.

### 5.2 Sensor placements

Our first example with real data is the sensor placements problem, where we try to select sensor locations to minimize the variance of observations. The dataset we used here is temperature measurements at discretized finite locations $V$ obtained using the NIMS sensor node deployed at a lake near the University of California, Merced [9, 12] ($|V| = 86$).[4] As in [12], we evaluated the set of locations $S \subseteq V$ using the averaged variance reduction $f(S) = Var(\emptyset) - Var(S) = \frac{1}{n}\sum_s F_s(S)$, where $F_s(S) = \sigma_s^2 - \sigma_{s|S}^2$ is the variance reduction and $\sigma_{s|S}^2$ denote the predictive variance at location $s \in V$ after observing locations $S \subseteq V$. This function is monotone and submodular. The graphs in Fig. 5 show the computation time of our algorithm, and the accuracy improvement of our calculated solution over that of the greedy algorithm (%), respectively, for $\epsilon = 0.05, 0.1, 0.2$. Both the computation time and improvement are large at around $k = 5$ compared with other choices of $k$. This is because the greedy solutions are good when $k$ is either very small or large.

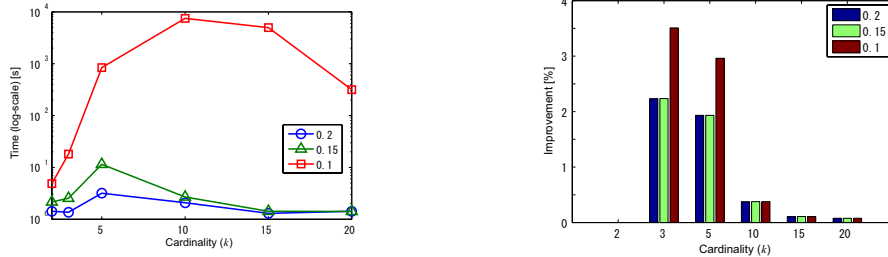

Figure 5: Computational time (left) and accuracy improvement over the greedy algorithm (right).

Table 1: Selected words with [the values of information gain, classification precision].

| $k$ | greedy | submodularity cuts |
|---|---|---|
| 5 | (tonn,'agricultur',trade,pct,'market')[2.59,0.53] | $\to$ ('week',tonn,trade,pct,'washington')[2.66,0.58] |
| 10 | ( . . .,week,oil,price,'dollar','offici')[3.55,0.57] | $\to$ ( . . .,price,oil,'bank','produc','blah')[3.88,0.62] |

## 5.3 Feature selection in text classification

Our second real test case is feature selection in document classification using the Reuters-21578 dataset. We applied the greedy and submodularity cuts algorithms to the training set that includes 7,770 documents with 5,180 words (features) and 90 categories, where we used the information gain as a criterion [4]. Table 1 shows the selected words by the algorithms in the cases of $k = 5, 10$ (for the proposed algorithm $\epsilon = 0.003$ in both cases) with the values of information gain and classification precision ($tp/(tp + fp)$, $tp$; true positive, $fp$; false positive). For classification on the test set (3,019 documents with 5,180 words and 90 categories), we applied a Naive Bayes classifier with the selected features. The submodularity cuts algorithm selected several different words from that of the greedy algorithm. We can see that the words selected by our algorithm would have high predictive power even though the number of the chosen words is very small.

## 6 Conclusions

In this paper, we presented a cutting-plane algorithm for submodular maximization problems, which can be implemented as an iterative binary-integer linear programming procedure. We derived a cutting plane procedure, called the submodularity cut, based on the submodularity of a set function through the Lovász extension, and showed this cut assures that the algorithm converges to the optimum in finite iterations. Moreover, we presented a way to evaluate an upper bound of the optimal value with the help of Nemhauser and Wolsey [17], which enables us to ensure the accuracy of the current best solution and to calculate an intended $\epsilon$-optimal solution for a predetermined $\epsilon > 0$. Our new algorithm computationally compared favorably against the existing algorithm on artificial datasets, and also showed improved performance on the real-world applications of sensor placements and feature selection in text classification.

The submodular maximization problem treated in this paper covers broad range of applications in machine learning. In future works, we will develop frameworks with $\epsilon$-optimality guarantees for more general problem settings such as knapsack constraints [21] and not nondecreasing submodular functions. This will be make the submodularity cuts framework applicable to a still wider variety of machine learning problems.

### Acknowledgments

This research was supported in part by JSPS Global COE program "Computationism as a Foundation for the Sciences", KAKENHI (20800019 and 21680025), the JFE 21st Century Foundation, and the Functional RNA Project of New Energy and Industrial Technology Development Organization (NEDO). Further support was received from a PASCAL2 grant, and by NSF grant IIS-0535100. Also, we are very grateful to the reviewers for helpful comments.

## Footnotes

[1] A submodular function is called nondecreasing if $f(A) \leq f(B)$ for $(A \subseteq B)$. For example, an entropy function is nondecreasing but a cut function on nodes is not.

[2]For a submodular function, the Lovász extension (2) is known to be equal to

[4]The covariance matrix of the Gaussian process that models the measurements is available in Matlab Toolbox for Submodular Function Optimization (http://www.cs.caltech.edu/~krausea/sfo/).

# References

[1] A. Das and D. Kempe. Algorithms for subset selection in linear regression. In R. E. Ladner and C. Dwork, editors, *Proc. of the 40th Annual ACM Symp. on Theory of Computing (STOC 2008)*, pages 45–54, 2008.

[2] J. Edmonds. Submodular functions, matroids, and certain polyhedra. In R. Guy, H. Hanani, N. Sauer, and J. Shönheim, editors, *Combinatorial Structures and Their Applications*, pages 69–87. Gordon and Breach, 1970.

[3] U. Feige. A threshold of ln n for approximating set cover. *Journal of the ACM*, 45:634–652, 1998.

[4] G. Forman. An extensive empirical study of feature selection metrics for text classification. *Journal of Machine Learning Research*, 3:1289–1305, 2003.

[5] S. Fujishige. *Submodular Functions and Optimization*. Elsevier, second edition, 2005.

[6] F. Glover. Convexity cuts and cut search. *Operations Research*, 21:123–134, 1973.

[7] F. Glover. Polyhedral convexity cuts and negative edge extension. *Zeitschrift für Operations Research*, 18:181–186, 1974.

[8] B. Goldengorin. Maximization of submodular functions: Theory and enumeration algorithms. *European Journal of Operational Research*, 198(1):102–112, 2009.

[9] T. C. Harmon, R. F. Ambrose, R. M. Gilbert, J. C. Fisher, M. Stealey, and W. J. Kaiser. High resolution river hydraulic and water quality characterization using rapidly deployable. Technical report, CENS,, 2006.

[10] S. C. H. Hoi, R. Jin, J. Zhu, and M. R. Lyu. Batch mode active learning and its application to medical image classification. In *Proc. of the 23rd int'l conf. on Machine learning (ICML 2006)*, pages 417–424, 2006.

[11] R. Horst and H. Tuy. *Global Optimization (Deterministic Approaches)*. Springer, 3 edition, 1996.

[12] A. Krause, H. B. McMahan, C. Guestrin, and A. Gupta. Robust submodular observation selection. *Journal of Machine Learning Research*, 9:2761–2801, 2008.

[13] A. Krause, A. Singh, and C. Guestrin. Near-optimal sensor placements in Gaussian processes: Theory, efficient algorithms and empirical studies. *Journal of Machine Learning Research*, 9:235–284, 2009.

[14] H. Lee, G. L. Nemhauser, and Y. Wang. Maximizing a submodular function by integer programming: Polyhedral results for the quadratic case. *European Journal of Operational Research*, 94:154–166, 1996.

[15] L. Lovász. Submodular functions and convexity. In A. Bachem, M. Grötschel, and B. Korte, editors, *Mathematical Programming – The State of the Art*, pages 235–257. 1983.

[16] K. Murota. *Discrete Convex Analysis*, volume 10 of *Monographs on Discrete Math and Applications*. Society for Industrial & Applied, 2000.

[17] G. L. Nemhauser and L. A. Wolsey. Maximizing submodular set functions: formulations and analysis of algorithms. In P. Hansen, editor, *Studies on Graphs and Discrete Programming*, volume 11 of *Annals of Discrete Mathematics*. 1981.

[18] G. L. Nemhauser and L. A. Wolsey. *Integer and Combinatorial Optimization*. Wiley-Interscience, 1988.

[19] G. L. Nemhauser, L. A. Wolsey, and M. L. Fisher. An analysis of approximations for maximizing for submodular set functions – I. *Mathematical Programming*, 14:265–294, 1978.

[20] M. Porembski. Finitely convergent cutting planes for concave minimization. *Journal of Global Optimization*, 20(2):109–132, 2001.

[21] M. Sviridenko. A note on maximizing a submodular set function subject to a knapsack constraint. *Operations Research Letters*, 32(1):41–43, 2004.

[22] M. Thoma, H. Cheng, A. Gretton, J. Han, H. P. Kriegel, A. J. Smola, L. Song, P. S. Yu, X. Yan, and K. M. Borgwardt. Near-optimal supervised feature selection among frequent subgraphs. In *Proc. of the 2009 SIAM Conference on Data Mining (SDM 2009)*, pages 1075–1086, 2009.

[23] H. Tuy. Concave programming under linear constraints. *Soviet Mathematics Doklady*, 5:1437–1440, 1964.

